# FACTORIE: Probabilistic Programming via Imperatively Defined Factor Graphs

**Andrew McCallum, Karl Schultz, Sameer Singh**
Department of Computer Science
University of Massachusetts Amherst
Amherst, MA 01003
*{mccallum, kschultz, sameer}@cs.umass.edu*

## Abstract

Discriminatively trained undirected graphical models have had wide empirical success, and there has been increasing interest in toolkits that ease their application to complex relational data. The power in relational models is in their repeated structure and tied parameters; at issue is how to define these structures in a powerful and flexible way. Rather than using a declarative language, such as SQL or first-order logic, we advocate using an imperative language to express various aspects of model structure, inference, and learning. By combining the traditional, declarative, statistical semantics of factor graphs with imperative definitions of their construction and operation, we allow the user to mix declarative and procedural domain knowledge, and also gain significant efficiencies. We have implemented such *imperatively defined factor graphs* in a system we call FACTORIE, a software library for an object-oriented, strongly-typed, functional language. In experimental comparisons to Markov Logic Networks on joint segmentation and coreference, we find our approach to be 3-15 times faster while reducing error by 20-25%—achieving a new state of the art.

## 1   Introduction

Conditional random fields [1], or discriminatively trained undirected graphical models, have become the tool of choice for addressing many important tasks across bioinformatics, natural language processing, robotics, and many other fields [2, 3, 4]. While relatively simple structures such as linear chains, grids, or fully-connected affinity graphs have been employed successfully in many contexts, there has been increasing interest in more complex relational structure—capturing more arbitrary dependencies among sets of variables, in repeated patterns—and interest in models whose variable-factor structure changes during inference, as in parse trees and identity uncertainty. Implementing such complex models from scratch in traditional programming languages is difficult and error-prone, and hence there has been several efforts to provide a high-level language in which models can be specified and run. For generative, directed graphical models these include BLOG [5], IBAL [6], and Church [7]. For conditional, undirected graphical models, these include Relational Markov Networks (RMNs) using SQL [8], and Markov Logic Networks (MLNs) using first-order logic [9].

Regarding logic, for many years there has been considerable effort in integrating first-order logic and probability [9, 10, 11, 12, 13]. However, we contend that in many of these proposed combinations, the 'logic' aspect is not crucial to the ultimate goal of accurate and expressive modeling. The power of relational factor graphs is in their repeated relational structure and tied parameters. First-order logic is one way to specify this repeated structure, but it is less than ideal because of its focus on boolean outcomes and inability to easily and efficiently express relations such as graph reachability and set size comparison. Logical inference is used in some of these systems, such as PRISM [12], but in others, such as Markov Logic [9], it is largely replaced by probabilistic inference.

This paper proposes an approach to probabilistic programming that preserves the *declarative* statistical semantics of factor graphs, while at the same time leveraging *imperative* constructs (pieces of procedural programming) to greatly aid both efficiency and natural intuition in specifying model structure, inference, and learning, as detailed below. Our approach thus supports users in combining both declarative and procedural knowledge. Rather than first-order logic, model authors have access to a Turing complete language when writing their model specification. The point, however, is not merely to have greater formal expressiveness; it is ease-of-use and efficiency.

We term our approach *imperatively defined factor graphs* (IDFs). Below we develop this approach in the context of Markov chain Monte-Carlo inference, and define four key imperative constructs—arguing that they provide a natural interface to central operations in factor graph construction and inference. These imperative constructs (1) define the structure connecting variables and factors, (2) coordinate variable values, (3) map the variables neighboring a factor to sufficient statistics, and (4) propose jumps from one possible world to another. A model written as an IDF *is* a factor graph, with all the traditional semantics of factors, variables, possible worlds, scores, and partition functions; we are simply providing an extremely flexible language for their succinct specification, which also enables efficient inference and learning.

Our first embodiment of the approach is the system we call FACTORIE (loosely named for "Factor graphs, Imperative, Extensible", see `http://factorie.cs.umass.edu`) strongly-typed, functional programming language *Scala* [14]. The choice of *Scala* stems from key inherent advantages of the language itself, plus its full interoperability with *Java*, and recent growing usage in the machine learning community. By providing a library and direct access to a full programming language (as opposed to our own, new "little language"), the model authors have familiar and extensive resources for implementing the procedural aspects of the design, as well as the ability to beneficially mix data pre-processing, evaluation, and other book-keeping code in the same files as the probabilistic model specification. Furthermore, FACTORIE is object-oriented in that variables and factor templates are *objects*, supporting inheritance, polymophism, composition, and encapsulation.

The contributions of this paper are introducing the novel IDF methodology for specifying factor graphs, and successfully demonstrating it on a non-trivial task. We present experimental results applying FACTORIE to the substantial task of joint inference in segmentation and coreference of research paper citations, surpassing previous state-of-the-art results. In comparison to Markov Logic (Alchemy) on the same data, we achieve a 20-25% reduction in error, and do so 3-15 times faster.

## 2 Imperatively Defined Factor Graphs

A factor graph $G$ is a bipartite graph over factors and variables defining a probability distribution over a set of target variables $\mathbf{y}$, optionally conditioned on observed variables $\mathbf{x}$. A factor $\Psi_i$ computes a scalar value over the subset of variables that are its neighbors in the graph. Often this real-valued function is defined as the exponential of the dot product over sufficient statistics $\{f_{ik}(\mathbf{x}_i, \mathbf{y}_i)\}$ and parameters $\{\theta_{ik}\}$, where $k \in \{1 \ldots K_i\}$ and $K_i$ is the number of parameters for factor $\Psi_i$.

Factor graphs often use *parameter tying*, *i.e.* the same parameters for several factors. A *factor template* $T_j$ consists of parameters $\{\theta_{jk}\}$, sufficient statistic functions $\{f_{jk}\}$, and a description of an arbitrary relationship between variables, yielding a set of satisfying tuples $\{(\mathbf{x}_i, \mathbf{y}_i)\}$. For each of these variable tuples $(\mathbf{x}_i, \mathbf{y}_i) \in T_j$ that fulfills the relationship, the factor template instantiates a factor that shares $\{\theta_{jk}\}$ and $\{f_{jk}\}$ with all other instantiations of $T_j$. Let $\mathcal{T}$ be the set of factor templates. In this case the probability distribution is defined:

$$p(\mathbf{y}|\mathbf{x}) = \frac{1}{Z(\mathbf{x})} \prod_{T_j \in \mathcal{T}} \prod_{(\mathbf{x}_i, \mathbf{y}_i) \in T_j} \exp\left[\sum_{k=1}^{K_j} \theta_{jk} f_{jk}(\mathbf{x}_i, \mathbf{y}_i)\right].$$

As in all relational factor graphs, our language supports variables and factor template definitions. In our case the variables—which can be binary, categorical, ordinal, real, etc—are typed objects in the object-oriented language, and can be sub-classed. Relations between variables can be represented directly as members (instance variables) in these variable objects, rather than as indices into global tables. In addition we allow for new variable types to be programmed by model authors via polymorphism. For example, the user can easily create new variable types such as a set-valued variable type,

Figure 1: Example of variable classes for a linear chain and a coreference model.

```scala
class Token(str:String) extends EnumVariable(str)
class Label(str:String, val token:Token) extends EnumVariable(str) with VarInSeq
class Mention(val string:String) extends PrimitiveVariable[Entity]
class Entity extends SetVariable[Mention] {
   var canonical:String = ""
   def add(m:Mention, d:DiffList) = {
      super.add(m,d); m.set(this,d)
      canonical = recomputeCanonical(members)
   }
   def remove(m:Mention, d:DiffList) = {
      super.remove(m,d); m.set(null,d)
      canonical = recomputeCanonical(members)
   }
}
```

representing a group of unique values, as well as traits augmenting variables to represent sequences of elements with left and right neighbors.

Typically, IDF programming consists of two distinct *stages*: defining the data representation, then defining the factors for scoring. This separation offers great flexibility. In the first stage the model author implements infrastructure for storing a possible world—variables, their relations and values. Somewhat surprisingly, authors can do this with a mind-set and style they would employ for deterministic programming, including usage of standard data structures such as linked lists, hash tables and objects embedded in other objects. In some cases authors must provide API functions for "undoing" and "redoing" changes to variables that will be tracked by MCMC, but in most cases such functionality is already provided by the library's wide variety of variable object implementations. For example, in a linear-chain CRF model, a variable containing a word token can be declared as the Token class shown in Figure 1.[1] A variable for labels can be declared similarly, with the addition that each Label[2] object has an instance variable that points to its corresponding Token. The second stage of our linear-chain CRF implementation is described in Section 2.2.

Consider also the task of entity resolution in which we have a set of Mentions to be co-referenced into Entities. A Mention contains its string form, but its value as a random variable is the Entity to which it is currently assigned. An Entity is a set-valued variable—the set of Mentions assigned to it; it holds and maintains a canonical string form representative of all its Mentions (see Figure 1[3]). The add/remove methods are explained in section 2.3.

## 2.1 Inference and Imperative Constraint Preservation

For inference, we rely on MCMC to achieve efficiency with models that not only have large tree-width but an exponentially-sized unrolled network, as is common with complex relational data [15, 9, 5]. The key is to avoid unrolling the network over multiple hypotheses, and to represent only one variable-value configuration at a time. As in BLOG [5], MCMC steps can adjust model structure as necessary, and with each step the FACTORIE library automatically builds a *DiffList*—a compact object containing the variables changed by the step, as well as undo and redo capabilities. Calculating the factor graph's 'score' for a step only requires *DiffList* variables, their factors, and neighboring variables, as described in Section 2.4. In fact, unlike BLOG and BLAISE [16], we build inference and learning entirely on *DiffList* scores and never need to score the entire model. This enables efficient reasoning about observed data larger than memory, or models in which the number of factors is a high-degree polynomial of the number of variables.

A key component of many MCMC inference procedures is the *proposal distribution* that proposes changes to the current configuration. This is a natural place for injecting prior knowledge about coordination of variable values and various structural changes. In fact, in some cases we can avoid

Figure 2: Examples of FACTORIE factor templates. Some error-checking code is elided for brevity.

```
val crfTemplate = new TemplateWithDotStatistics3[Label,Label,Token] {
    def unroll1 (label:Label) = Factor(label, label.next, label.token)
    def unroll2 (label:Label) = Factor(label.prev, label, label.prev.token)
    def unroll3 (token:Token) = throw new Error("Token values shouldn't change")
}
val depParseTemplate = new Template1[Node] with DotStatistics2[Word,Word] {
    def unroll1(n:Node) = n.selfAndDescendants
    def statistics(n:Node) = Stat(n.word, closestVerb(n).word)
    def closestVerb(n:Node) = if (isVerb(n.word)) n else closestVerb(n.parent)
}
val corefTemplate = new Template2[Mention,Entity] with DotStatistics1[Bool] {
    def unroll1 (m:Mention) = Factor(m, m.entity)
    def unroll2 (e:Entity) = for (mention <- e.mentions) yield Factor(mention, e)
    def statistics(m:Mention,e:Entity) = Bool(distance(m.string,e.canonical)<0.5)
}
val logicTemplate1 = Forany[Person] { p => p.smokes --> p.cancer }
val logicTemplate2 = Forany[Person] { p => p.friends.smokes <--> p.smokes }
```

expensive deterministic factors altogether with *property-preserving* proposal functions [17]. For example, coreference transitivity can be efficiently enforced by proper initialization and a transitivity-preserving proposal function; projectivity in dependency parsers can be enforced similarly. We term this *imperative constraint preservation*. In FACTORIE proposal distributions may be implemented by the model author. Alternatively, the FACTORIE library provides several default inference methods, including Gibbs sampling, as well as default proposers for many variable classes.

## 2.2 Imperative Structure Definition

At the heart of model structure definition is the pattern of connectivity between variables and factors, and the *DiffList* must have extremely efficient access to this. Unlike BLOG, which uses a complex, highly-indexed data structure that must be updated during inference, we instead specify this connectivity imperatively: factor template objects have methods (*e.g.*, unroll1, unroll2, *etc.*, one for each factor argument) that find the factor's other variable neighbors given a single variable from the *DiffList*. This is typically accomplished using a simple data structure that is already available as part of the natural representation of the data, (*e.g.*, as would be used by a non-probabilistic programmer). The unroll method then constructs a Factor with these neighbors as arguments, and returns it. The unroll method may optionally return multiple Factors in response to a single changed variable. Note that this approach also efficiently supports a model structure that varies conditioned on variable values, because the unroll methods can examine and perform calculations on these values.

Thus we now have the second stage of FACTORIE programming, in which the model author implements the factor templates that define the factors which score possible worlds. In our linear-chain CRF example, the factor between two succesive Labels and a Token might be declared as crfTemplate in Figure 2. Here unroll1 simply uses the token instance variable of each Label to find the corresponding third argument to the factor. This simple example does not, however, show the true expressive power of *imperative structure definition*. Consider instead a model for dependency parsing (with similarly defined Word and Node variables). In the same Figure, depParsingTemplate defines a template for factors that measure compatibility between a word and its closest verb as measured through parse tree connectivity. Such arbitrary-depth graph search is awkward in first-order logic, yet it is a simple one-line recursive method in FACTORIE. The statistics method is described below in Section 2.4.

Consider also the coreference template measuring the compatibility between a Mention and the canonical representation of its assigned Entity. In response to a moved Mention, unroll1 returns a factor between the Mention and its newly assigned Entity. In response to a changed Entity, unroll2 returns a list of factors between itself all its member Mentions. It is inherent that sometimes different unroll methods will construct multiple copies of the same factor; they are automatically deduplicated by the FACTORIE library. Syntactic sugar for extended first-order logic primitives is also provided, and these can be mixed with imperative constructs; see the bottom of Figure 2 for two small examples. Specifying templates in FACTORIE can certainly be more verbose when not restricted to first-order logic; in this case we trade off some brevity for flexibility.

## 2.3 Imperative Variable Coordination

Variables' `value-assignment` methods can be overriden to automatically change other variable values in coordination with the assignment—an often-desirable encapsulation of domain knowledge we term *imperative variable coordination*. For example, in response to a named entity label change, a coreference mention can have its string value automatically adjusted, rather than relying on MCMC inference to stumble upon this self-evident coordination. In Figure 1, `Entity` does a basic form of coordination by re-calculating its canonical string representation whenever a `Mention` is added or removed from its set.

The ability to use prior knowledge for *imperative variable coordination* also allows the designer to define the feasible region for the sampling. In the proposal function, users make changes by calling `value-assignment` functions, and any changes made automatically through coordinating variables are appended to the *DiffList*. Since a factor template's contribution to the overall score will not change unless its neighboring variables have changed, once we know every variable that has changed we can efficiently score the proposal.

## 2.4 Imperative Variable-Statistics Mapping

In a somewhat unconventional use of functional mapping, we support a separation between factor *neighbors* and *sufficient statistics*. Neighbors are variables touching the factor whose changes imply that the factor needs to be re-scored. Sufficient statistics are the minimal set of variable values that determine the score contribution of the factor. These are usually the same; however, by allowing a function to perform the mapping, we provide an extremely powerful yet simple way to allow model designers to represent their data in natural ways, and concern themselves separately with how to parameterize them. For example, the two neighbors of a skip-edge factor [18] may each have cardinality equal to the number of named entities types, but we may only care to have the skip-edge factor enforce whether or not they match. We term this *imperative variable-statistics mapping*.

Consider `corefTemplate` in Figure 2, the neighbors of the template are ⟨`Mention`, `Entity`⟩ pairs. However, the sufficient statistic is simply a Boolean based on the "distance" of the unrolled `Mention` from the canonical value of the `Entity`. This allows the template to separate the natural representation of possible worlds from the sufficient statistics needed to score its factors. Note that these sufficient statistics can be calculated as arbitrary functions of the unrolled `Mention` and the `Entity`. The models described in Section 3 use a number of factors whose sufficient statistics derive from the domains of its neighbors as well as those with arbitrary feature functions based on their neighbors.

An MCMC proposal is scored as follows. First, a sample is generated from the proposal distribution, placing an initial set of variables in the *DiffList*. Next the `value-assignment` method is called for each of the variables on the *DiffList*, and via *imperative variable coordination* other variables may be added to the *DiffList*. Given the set of variables that have changed, FACTORIE iterates over each one and calls the `unroll` function for factor templates matching the variable's type. This dynamically provides the relevant structure of the graph via *imperative structure definition*, resulting in a set of factors that should be re-scored. The neighbors of each returned factor are given to the template's `statistics` function, and the sufficient statistics are used to generate the factor's score using the template's current parameter vector. These scores are summed, producing the final score for the MCMC step.

## 2.5 Learning

Maximum likelihood parameter estimation traditionally involves finding the gradient, however for complex models this can be prohibitively expensive since it requires the inference of marginal distributions over factors. Alternatively some have proposed online methods, such as perceptron, which avoids the need for marginals however still requires full decoding which can also be computationally expensive. We avoid both of these issues by using *sample-rank* [19]. This is a parameter estimation method that learns a ranking over all possible configurations by observing the difference between scores of proposed MCMC jumps. Parameter changes are made when the model's ranking of a proposed jump disagrees with a ranking determined by labeled truth. When there is such a disagreement, a perceptron-style update to active parameters is performed by finding all factors whose score has changed (i.e., factors with a neighbor in the *DiffList*). The active parameters are indexed by the

sufficient statistics of these factors. Sample-rank is described in detail in [20]. As with inference, learning is efficient because it uses the *DiffList* and the imperative constructs described earlier.

# 3   Joint Segmentation and Coreference

Tasks involving multiple information extraction steps are traditionally solved using a pipeline architecture, in which the output predictions of one stage are input to the next stage. This architecture is susceptible to cascading of errors from one stage to the next. To minimize this error, there has been significant interest in *joint inference* over multiple steps of an information processing pipeline [21, 22, 23]. Full joint inference usually results in exponentially large models for which learning and inference become intractable. One widely studied joint-inference task in information extraction is segmentation and coreference of research paper citation strings [21, 23, 24]. This involves segmenting citation strings into *author*, *title* and *venue* fields (*segmentation*), and clustering the citations that refer to the same underlying paper entity (*coreference*). Previous results have shown that joint inference reduces error [21], and this task provides a good testbed for probabilistic programming. We now describe an IDF for the task. For more details, see [24].

## 3.1   Variables and Proposal Distribution

As in the example given in Section 2, a `Mention` represents a citation and is a random variable that takes a single `Entity` as its value. An `Entity` is a set-valued variable containing `Mention` variables. This representation eliminates the need for an explicit transitivity constraint, since a `Mention` can hold only one `Entity` value, and this value is coordinated with the `Entity`'s set-value.

Variables for segmentation consist of `Tokens`, `Labels` and `Fields`. Each `Token` represents an observed word in a citation. Each `Token` has a corresponding `Label` which is an unobserved variable that can take one of four values: *author*, *title*, *venue* or *none*. There are three `Field` variables associated with each `Mention`, one for each field type (*author*, *venue* or *title*), that store the contiguous block of `Tokens` representing the `Field`; `Labels` and `Fields` are coordinated. This alternate representation of segmentation provides flexibility in specifying factor templates over predicted `Fields`.

The proposal function for coreference randomly selects a `Mention`, and with probability 0.8 moves it to a random existing cluster, otherwise to a new singleton cluster. The proposal function for segmentation selects a random `Field` and grows or shrinks it by a random amount. When jointly performing both tasks, one of the proposal functions is randomly selected. The `value-assignment` function for the `Field` ensures that the `Labels` corresponding to the affected `Tokens` are correctly set when a `Field` is changed. This is an example of *imperative variable coordination*.

## 3.2   Factor Templates

**Segmentation Templates:**   Segmentation templates examine only `Field`, `Label` and `Token` variables, *i.e.* not using information from coreference predictions. These factor templates are IDF translations of the Markov logic rules described in [21]. There is a template between every `Token` and its `Label`. Markov dependencies are captured by a template that examines successive `Labels` as well as the `Token` of the earlier `Label`. The sufficient statistics for these factors are the tuples created from the neighbors of the factor: *e.g.*, the values of two `Labels` and one `Token`. We also have a factor template examining every `Field` with features based on the presence of numbers, dates, and punctuation. This takes advantage of *variable-statistics mapping*.

**Coreference Templates:**   The isolated coreference factor templates use only `Mention` variables. They consist of two factor templates that share the same sufficient statistics, but have separate weight vectors and different ways of unrolling the graph. An *Affinity* factor is created for all pairs of `Mentions` that are coreferent, while a *Repulsion* factor is created for all pairs that are not coreferent. The features of these templates correspond to the *SimilarTitle* and *SimilarVenue* first-order features in [21]. We also add *SimilarDate* and *DissimilarDate* features that look at the "date-like" tokens.

**Joint Templates:**   To allow the tasks to influence each other, factor templates are added that are unrolled during both segmentation and coreference sampling. Thus these factor templates neighbor `Mentions`, `Fields`, and `Labels`, and use the segmentation predictions for coreference, and vice-versa. We add templates for the *JntInfCandidates* rule from [21]. We create this factor template

Table 1: Cora coreference and segmentation results

| | Coreference | | | Segmentation F1 | | | |
|---|---|---|---|---|---|---|---|
| | **Prec/Recall** | **F1** | **Cluster Rec.** | **Author** | **Title** | **Venue** | **Total** |
| Fellegi-Sunter | 78.0/97.7 | 86.7 | 62.7 | n/a | n/a | n/a | n/a |
| Isolated MLN | 94.3/97.0 | 95.6 | 78.1 | 99.3 | 97.3 | 98.2 | 98.2 |
| Joint MLN | 94.3/97.0 | 95.6 | 75.2 | 99.5 | 97.6 | 98.3 | 98.4 |
| Isolated IDF | 97.09/95.42 | **96.22** | **86.01** | 99.35 | 97.63 | 98.58 | **98.51** |
| Joint IDF | 95.34/98.25 | **96.71** | **94.62** | 99.42 | 97.99 | 98.78 | **98.72** |

such that $(m, m')$ are unrolled only if they are in the same `Entity`. The neighbors include `Label` and `Mention`. *Affinity* and *Repulsion* factor templates are also created between pairs of `Fields` of the same type; for *Affinity* the `Fields` belong to coreferent mention pairs, and for *Repulsion* they belong to a pair of mentions that are not coreferent. The features of these templates denote similarity between field strings, namely: *StringMatch, SubString, Prefix/SuffixMatch, TokenIntersectSize*, etc.

One notable difference between the *JntInfCandidate* and joint *Affinity/Repulsion* templates is the possible number of instantiations. *JntInfCandidates* can be calculated during preprocessing as there are $O(nm^2)$ of these (where $n$ is the maximum mention length, and $m$ is the number of mentions). However, preprocessing joint *Affinity/Repulsion* templates is intractable as the number of such factors is $O(m^2 n^4)$. We are able to deal with such a large set of possible factor instantiations due to the interplay of *structure definition*, *variable-statistics mapping*, and on-the-fly feature calculation.

Our model also contains a number of factor templates that cannot be easily captured by first-order logic. For example consider *StringMatch* and *SubString* between two fields. For arbitrary length strings these features require the model designer to specify convoluted logic rules. The rules are even less intuitive when considering a feature based on more complex calculations such as *StringEditDistance*. It is conceivable to preprocess and store all instantiations of these features, but in practice this is intractable. Thus on-the-fly feature calculation within FACTORIE is employed to remain tractable.

## 4 Experimental Results

The joint segmentation and coreference model described above is applied to the Cora dataset[25].[4] The dataset contains 1295 total mentions in 134 clusters, with a total of 36487 tokens. Isolated training consists of 5 loops of 100,000 samples each, and 300,000 samples for inference. For the joint task we run training for 5 loops of 250,000 samples each, with 750,000 samples for inference. We average the results of 10 runs of three-fold cross validation, with the same folds as [21]. Segmentation is evaluated on token precision, recall and F1. For coreference, pairwise coreference decisions are evaluated. The fraction of clusters that are correctly predicted (cluster recall) is also calculated.

In Table 1, we see both our isolated and joint models outperform the previous state-of-the-art results of [21] on both tasks. We see a $25.23\%$ error reduction in pairwise coreference F1, and a $20.0\%$ error reduction of tokenwise segmentation F1 when comparing to the joint MLN. The improvements of joint over isolated IDF are statistically significant at $1\%$ using the T-test.

The experiments run very quickly, which can be attributed to *sample-rank* and the application of *variable coordination* and *structure definition* of the models as described earlier. Each of the isolated tasks finishes initialization, training and evaluation within 3 minutes, while the joint task takes 18 minutes. The running time for the MLNs reported in [21] are between 50-90 minutes for learning and inference. Thus we can see that IDFs provide a significant boost in efficiency by avoiding the need to unroll or score the entire graph. Note also that the timing result from [21] is for a model that did not enforce transitivity constraints on the coreference predictions. Adding transitivity constraints dramatically increases running time [26], whereas the IDF supports transitivity implicitly.

# 5 Related Work

Over the years there have been many efforts to build graphical models toolkits. Many of them are useful as teaching aids, such as the Bayes Net Toolbox and Probabilistic Modeling Toolkit (PMTK)[27] (both in Matlab), but do not scale up to substantial real problems.

There has been growing interest in building systems that can perform as workhorses, doing real work on large data. For example, Infer.NET (CSoft) [28] is intended to be deployed in a number of Microsoft products, and has been applied to problems in computer vision. Like IDFs it is embedded in a pre-existing programming language, rather than embodying its own new "little language," and its users have commented positively about this facet. Unlike IDFs it is designed for messaging-passing inference, and must unroll the graphical model before inference, creating factors to represent all possible worlds, which makes it unsuitable for our applications. The very recent language Figaro [29] is also implemented as a library. Like FACTORIE it is implemented in Scala, and provides an object-oriented framework for models; unlike FACTORIE it tightly intertwines data representation and scoring, and it is not designed for changing model structure during inference; it also does not yet support learning.

BLOG [5] and some of its derivatives can also scale to substantial data sets, and, like IDFs, are designed for graphical models that cannot be fully unrolled. Unlike IDFs, BLOG, as well as IBAL [6] and Church [7], are designed for generative models, though Church can also represent conditional, undirected models. We are most interested in supporting advanced discriminative models of the type that have been successful for natural language processing, computer vision, bioinformatics, and elsewhere. Note that FACTORIE also supports generative models; for example latent Dirichlet allocation can be coded in about 15 lines.

Two systems focussing on discriminatively-trained relational models are relational Markov networks (RMNs) [8], and Markov logic networks (MLNs, with *Alchemy* as its most popular implementation). To define repeated relational structure and parameter tying, both use declarative languages: RMNs use SQL and MLNs use first-order logic. By contrast, as discussed above, IDFs are in essence an experiment in taking an imperative approach.

There has, however, been both historical and recently growing interest in using imperative programming languages for defining learning systems and probabilistic models. For example, work on theory refinement [30] viewed domain theories as "statements in a procedural programming language, rather than the common view of a domain theory being a collection of declarative Prolog statements." More recently, IBAL [6] and Church [7] are both fundamentally *programs* that describe the generative storyline for the data. IDFs, of course, share the combination of imperative programming with probabilistic modeling, but IDFs have their semantics defined by undirected factor graphs, and are typically discriminatively trained.

# 6 Conclusion

In this paper we have described imperatively defined factor graphs (IDFs), a framework to support efficient learning and inference in large factor graphs of changing structure. We preserve the traditional, declarative, statistical semantics of factor graphs while allowing imperative definitions of the model structure and operation. This allows model authors to combine both declarative and procedural domain knowledge, while also obtaining significantly more efficient inference and learning than declarative approaches. We have shown state-of-the-art results in citation matching that highlight the advantages afforded by IDFs for both accuracy and speed.

# Acknowledgments

This work was supported in part by NSF medium IIS-0803847; the Central Intelligence Agency, the National Security Agency and National Science Foundation under NSF grant IIS-0326249; SRI International subcontract #27-001338 and ARFL prime contract #FA8750-09-C-0181; Army prime contract number W911NF-07-1-0216 and University of Pennsylvania subaward number 103-548106. Any opinions, findings and conclusions or recommendations expressed in this material are the authors' and do not necessarily reflect those of the sponsor.

## Footnotes

[1]Objects of class EnumVariable hold variables with a value selected from a finite enumerated set.

[2]In *Scala* var/val indicates a variable declaration; trait VarInSeq provides methods for obtaining next and prev labels in a sequence.

[3]In *Scala* def indicates a function definition where the value returned is the last line-of-code in the function; members is the set of variables in the superclass SetVariable.

[4] Available at http://alchemy.cs.washington.edu/papers/poon07

# References

[1] John D. Lafferty, Andrew McCallum, and Fernando Pereira. Conditional random fields: Probabilistic models for segmenting and labeling sequence data. In *Int Conf on Machine Learning (ICML)*, 2001.

[2] Charles Sutton and Andrew McCallum. An introduction to conditional random fields for relational learning. In *Introduction to Statistical Relational Learning*. 2007.

[3] A. Bernal, K. Crammer, A. Hatzigeorgiou, and F. Pereira. Global discriminative learning for higher-accuracy computation gene prediction. In *PloS Computational Biology*, 2007.

[4] A. Quottoni, M. Collins, and T. Darrell. Conditional random fields for object recognition. In *NIPS*, 2004.

[5] Brian Milch. *Probabilistic Models with Unknown Objects*. PhD thesis, University of California, Berkeley, 2006.

[6] Avi Pfeffer. IBAL: A probabilistic rational programming language. In *IJCAI*, pages 733–740, 2001.

[7] Noah D. Goodman, Vikash K. Mansinghka, Daniel Roy, Keith Bonawitz, and Joshua B. Tenenbaum. Church: a language for generative models. In *Uncertainty in Artificial Intelligence (UAI)*, 2008.

[8] Ben Taskar, Abbeel Pieter, and Daphne Koller. Discriminative probabilistic models for relational data. In *Uncertainty in Artificial Intelligence (UAI)*, 2002.

[9] Matthew Richardson and Pedro Domingos. Markov logic networks. *Machine Learning*, 62(1-2), 2006.

[10] David Poole. Probabilistic horn abduction and bayesian networks. *Artificial Intelligence*, 64, 1993.

[11] Stephen Muggleton and Luc DeRaedt. Inductive logic programming theory and methods. In *Journal of Logic Programming*, 1994.

[12] Taisuke Sato and Yoshitaka Kameya. PRISM: a language for symbolic-statistical modeling. In *International Joint Conference on Artificial Intelligence (IJCAI)*, 1997.

[13] Luc De Raedt and Kristian Kersting. Probabilistic logic learning. *SIGKDD Explorations: Multi-Relational Data Mining*, 2003.

[14] Martin Odersky. An Overview of the Scala Programming Language (second edition). Technical Report IC/2006/001, EPFL Lausanne, Switzerland, 2006.

[15] Aron Culotta and Andrew McCallum. Tractable learning and inference with high-order representations. In *ICML WS on Open Problems in Statistical Relational Learning*, 2006.

[16] Keith A. Bonawitz. *Composable Probabilistic Inference with Blaise*. PhD thesis, MIT, 2008.

[17] Aron Culotta. *Learning and inference in weighted logic with application to natural language processing*. PhD thesis, University of Massachusetts, 2008.

[18] Charles Sutton and Andrew McCallum. Collective segmentation and labeling of distant entities in information extraction. Technical Report TR#04-49, University of Massachusetts, July 2004.

[19] Aron Culotta, Michael Wick, and Andrew McCallum. First-order probabilistic models for coreference resolution. In *NAACL: Human Language Technologies (NAACL/HLT)*, 2007.

[20] Khashayar Rohanimanesh, Michael Wick, and Andrew McCallum. Inference and learning in large factor graphs with a rank based objective. Technical Report UM-CS-2009-08, University of Massachusetts, Amherst, 2009.

[21] Hoifung Poon and Pedro Domingos. Joint inference in information extraction. In *AAAI*, 2007.

[22] Vasin Punyakanok, Dan Roth, and Wen-tau Yih. The necessity of syntactic parsing for semantic role labeling. In *International Joint Conf on Artificial Intelligence (IJCAI)*, pages 1117–1123, 2005.

[23] Ben Wellner, Andrew McCallum, Fuchun Peng, and Michael Hay. An integrated, conditional model of information extraction and coreference with application to citation matching. In *AUAI*, 2004.

[24] Sameer Singh, Karl Schultz, and Andrew McCallum. Bi-directional joint inference for entity resolution and segmentation using imperatively-defined factor graphs. In *ECML PKDD*, pages 414–429, 2009.

[25] Andrew McCallum, Kamal Nigam, Jason Rennie, and Kristie Seymore. A machine learning approach to building domain-specific search engines. In *Int Joint Conf on Artificial Intelligence (IJCAI)*, 1999.

[26] Hoifung Poon, Pedro Domingos, and Marc Sumner. A general method for reducing the complexity of relational inference and its application to MCMC. In *AAAI*, 2008.

[27] Kevin Murphy and Matt Dunham. PMTK: Probabilistic modeling toolkit. In *Neural Information Processing Systems (NIPS) Workshop on Probabilistic Programming*, 2008.

[28] John Winn and Tom Minka. Infer.NET/CSoft, 2008. http://research.microsoft.com/mlp/ml/Infer/Csoft.htm.

[29] Avi Pfeffer. Figaro: An Object-Oriented Probabilistic Programming Language. Technical report, Charles River Analytics, 2009.

[30] Richard Maclin and Jude W. Shavlik. Creating advice-taking reinforcement learners. *Machine Learning*, 22, 1996.

